# Learning the structure of manifolds using random projections

**Yoav Freund** *
UC San Diego

**Sanjoy Dasgupta** †
UC San Diego

**Mayank Kabra**
UC San Diego

**Nakul Verma**
UC San Diego

## Abstract

We present a simple variant of the $k$-d tree which automatically adapts to intrinsic low dimensional structure in data.

## 1 Introduction

The curse of dimensionality has traditionally been the bane of nonparametric statistics, as reflected for instance in convergence rates that are exponentially slow in dimension. An exciting way out of this impasse is the recent realization by the machine learning and statistics communities that in many real world problems the high dimensionality of the data is only superficial and does not represent the true complexity of the problem. In such cases data of low *intrinsic* dimension is embedded in a space of high *extrinsic* dimension.

For example, consider the representation of human motion generated by a motion capture system. Such systems typically track marks located on a tight-fitting body suit. The number of markers, say $N$, is set sufficiently large in order to get dense coverage of the body. A posture is represented by a $(3N)$-dimensional vector that gives the 3D location of each of the $N$ marks. However, despite this seeming high dimensionality, the number of degrees of freedom is relatively small, corresponding to the dozen-or-so joint angles in the body. The marker positions are more or less deterministic functions of these joint angles. Thus the data lie in $\mathbb{R}^{3N}$, but on (or very close to) a *manifold* [4] of small dimension.

In the last few years, there has been an explosion of research investigating methods for learning in the context of low-dimensional manifolds. Some of this work (for instance, [2]) exploits the low intrinsic dimension to improve the convergence rate of supervised learning algorithms. Other work (for instance, [12, 11, 1]) attempts to find an embedding of the data into a low-dimensional space, thus finding an explicit mapping that reduces the dimensionality.

In this paper, we describe a new way of modeling data that resides in $\mathbb{R}^D$ but has lower intrinsic dimension $d < D$. Unlike many manifold learning algorithms, we do not attempt to find a single unified mapping from $\mathbb{R}^D$ to $\mathbb{R}^d$. Instead, we hierarchically partition $\mathbb{R}^D$ into pieces in a manner that is provably sensitive to low-dimensional structure. We call this spatial data structure a *random projection tree* (RP tree). It can be thought of as a variant of the $k$-d tree that is provably manifold-adaptive.

### $k$-d trees, RP trees, and vector quantization

Recall that a $k$-d tree [3] partitions $\mathbb{R}^D$ into hyperrectangular cells. It is built in a recursive manner, splitting along one coordinate direction at a time. The succession of splits corresponds to a binary tree whose leaves contain the individual cells in $\mathbb{R}^D$. These trees are among the most widely-used methods for spatial partitioning in machine learning and computer vision.

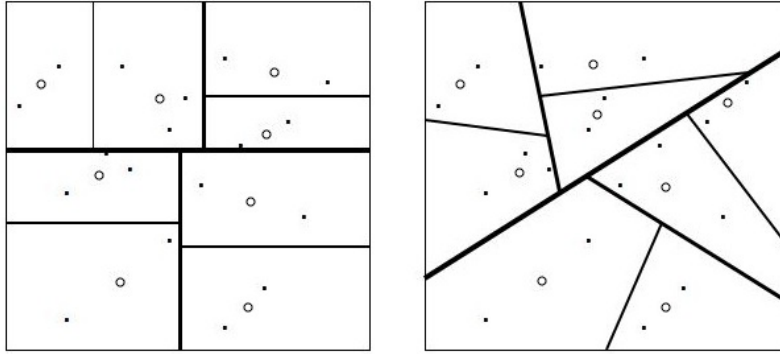

Figure 1: Left: A spatial partitioning of $\mathbb{R}^2$ induced by a $k$-d tree with three levels. The dots are data vectors; each circle represents the mean of the vectors in one cell. Right: Partitioning induced by an RP tree.

On the left part of Figure 1 we illustrate a $k$-d tree for a set of vectors in $\mathbb{R}^2$. The leaves of the tree partition $\mathbb{R}^D$ into *cells*; given a query point $q$, the cell containing $q$ is identified by traversing down the $k$-d tree. Each cell can be thought of as having a representative vector: its mean, depicted in the figure by a circle. The partitioning together with these mean vectors define a *vector quantization* (VQ) of $\mathbb{R}^2$: a mapping from $\mathbb{R}^2$ to a finite set of representative vectors (called a "codebook" in the context of lossy compression methods). A good property of this tree-structured vector quantization is that a vector can be mapped efficiently to its representative. The design goal of VQ is to minimize the error introduced by replacing vectors with their representative.

We quantify the VQ error by the average *squared* Euclidean distance between a vector in the set and the representative vector to which it is mapped. This error is closely related (in fact, proportional) to the *average diameter* of cells, that is, the average squared distance between pairs of points in a cell.[1] As the depth of the $k$-d tree increases the diameter of the cells decreases and so does the VQ error. However, in high dimension, the rate of decrease of the average diameter can be very slow. In fact, as we show in the supplementary material, there are data sets in $\mathbb{R}^D$ for which a $k$-d tree requires $D$ levels in order to halve the diameter. This slow rate of decrease of cell diameter is fine if $D = 2$ as in Figure 1, but it is disastrous if $D = 1000$. Constructing 1000 levels of the tree requires $2^{1000}$ data points! This problem is a real one that has been observed empirically: $k$-d trees are prone to a curse of dimensionality.

What if the data have low intrinsic dimension? In general, $k$-d trees will not be able to benefit from this; in fact the bad example mentioned above has intrinsic dimension $d = 1$. But we show that a simple variant of the $k$-d tree does indeed decrease cell diameters much more quickly. Instead of splitting along coordinate directions, we use randomly chosen unit vectors, and instead of splitting data exactly at the median, we use a more carefully chosen split point. We call the resulting data structure a *random projection tree* (Figure 1, right) and we show that it admits the following theoretical guarantee (formal statement is in the next section).

> Pick any cell $C$ in the RP tree, and suppose the data in $C$ have intrinsic dimension $d$. Pick a descendant cell $\geq d$ levels below; then with constant probability, this descendant has average diameter at most half that of $C$.[2]

There is no dependence at all on the extrinsic dimensionality ($D$) of the data. We thus have a vector quantization construction method for which the diameter of the cells depends on the intrinsic dimension, rather than the extrinsic dimension of the data.

A large part of the benefit of RP trees comes from the use of random unit directions, which is rather like running $k$-d trees with a preprocessing step in which the data are projected into a random

low-dimensional subspace. In fact, a recent experimental study of nearest neighbor algorithms [8] observes that a similar pre-processing step improves the performance of nearest neighbor schemes based on spatial data structures. Our work provides a theoretical explanation for this improvement and shows both theoretically and experimentally that this improvement is significant. The explanation we provide is based on the assumption that the data has low intrinsic dimension.

Another spatial data structure based on random projections is the *locality sensitive hashing* scheme [6].

**Manifold learning and near neighbor search**

The fast rate of diameter decrease in random projection trees has many consequences beyond the quality of vector quantization. In particular, the statistical theory of tree-based statistical estimators — whether used for classification or regression — is centered around the rate of diameter decrease; for details, see for instance Chapter 20 of [7]. Thus RP trees generically exhibit faster convergence in all these contexts.

Another case of interest is nearest neighbor classification. If the diameter of cells is small, then it is reasonable to classify a query point according to the majority label in its cell. It is not necessary to find the *nearest neighbor*; after all, the only thing special about this point is that it happens to be close to the query. The classical work of Cover and Hart [5] on the Bayes risk of nearest neighbor methods applies equally to the majority vote in a small enough cell.

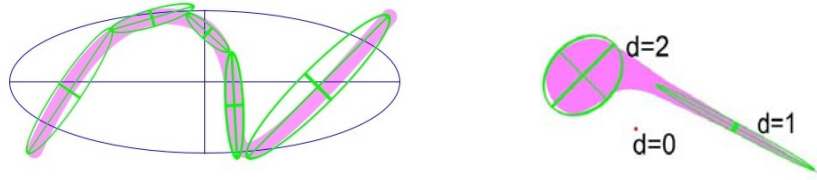

Figure 2: **Distributions with low intrinsic dimension.** The purple areas in these figures indicate regions in which the density of the data is significant, while the complementary white areas indicate areas where data density is very low. The left figure depicts data concentrated near a one-dimensional manifold. The ellipses represent mean+PCA approximations to subsets of the data. Our goal is to partition data into small diameter regions so that the data in each region is well-approximated by its mean+PCA. The right figure depicts a situation where the dimension of the data is variable. Some of the data lies close to a one-dimensional manifold, some of the data spans two dimensions, and some of the data (represented by the red dot) is concentrated around a single point (a zero-dimensional manifold).

Finally, we return to our original motivation: modeling data which lie close to a low-dimensional manifold. In the literature, the most common way to capture this manifold structure is to create a graph in which nodes represent data points and edges connect pairs of nearby points. While this is a natural representation, it does not scale well to very large datasets because the computation time of closest neighbors grows like the square of the size of the data set. Our approach is fundamentally different. Instead of a bottom-up strategy that starts with individual data points and links them together to form a graph, we use a top-down strategy that starts with the whole data set and partitions it, in a hierarchical manner, into regions of smaller and smaller diameter. Once these individual cells are small enough, the data in them can be well-approximated by an affine subspace, for instance that given by principal component analysis. In Figure 2 we show how data in two dimensions can be approximated by such a set of local ellipses.

## 2  The RP tree algorithm

### 2.1  Spatial data structures

In what follows, we assume the data lie in $\mathbb{R}^D$, and we consider spatial data structures built by recursive binary splits. They differ only in the nature of the split, which we define in a subroutine

called CHOOSERULE. The core tree-building algorithm is called MAKETREE, and takes as input a data set $S \subset \mathbb{R}^D$.

**procedure** MAKETREE($S$)
  **if** $|S| < MinSize$
    **then return** ($Leaf$)
    **else** $\begin{cases} Rule \leftarrow \text{CHOOSERULE}(S) \\ LeftTree \leftarrow \text{MAKETREE}(\{x \in S : Rule(x) = \text{true}\}) \\ RightTree \leftarrow \text{MAKETREE}(\{x \in S : Rule(x) = \text{false}\}) \\ \textbf{return } ([Rule, LeftTree, RightTree]) \end{cases}$

A natural way to try building a manifold-adaptive spatial data structure is to split each cell along its principal component direction (for instance, see [9]).

**procedure** CHOOSERULE($S$)
  **comment:** PCA tree version

  let $u$ be the principal eigenvector of the covariance of $S$
  $Rule(x) := x \cdot u \leq \text{median}(\{z \cdot u : z \in S\})$
  **return** ($Rule$)

This method will do a good job of adapting to low intrinsic dimension (details omitted). However, it has two significant drawbacks in practice. First, estimating the principal eigenvector requires a significant amount of data; recall that only about $1/2^k$ fraction of the data winds up at a cell at level $k$ of the tree. Second, when the extrinsic dimension is high, the amount of memory and computation required to compute the dot product between the data vectors and the eigenvectors becomes the dominant part of the computation. As each node in the tree is likely to have a different eigenvector this severely limits the feasible tree depth. We now show that using random projections overcomes these problems while maintaining the adaptivity to low intrinsic dimension.

## 2.2 Random projection trees

We shall see that the key benefits of PCA-based splits can be realized much more simply, by picking *random* directions. To see this pictorially, consider data that is concentrated on a subspace, as in the following figure. PCA will of course correctly identify this subspace, and a split along the principal eigenvector $u$ will do a good job of reducing the diameter of the data. But a random direction $v$ will also have some component in the direction of $u$, and splitting along the median of $v$ will not be all that different from splitting along $u$.

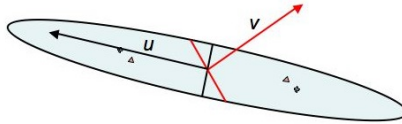

Figure 3: **Intuition:** a random direction is almost as good as the principal eigenvector.

Now only medians need to be estimated, not principal eigenvectors; this significantly reduces the data requirements. Also, we can use the same random projection in different places in the tree; all we need is to choose a large enough set of projections that, with high probability, there is be a good projection direction for each node in the tree. In our experience setting the number of projections equal to the depth of the tree is sufficient. Thus, for a tree of depth $k$, we use only $k$ projection vectors $v$, as opposed to $2^k$ with a PCA tree. When preparing data to train a tree we can compute the $k$ projection values before building the tree. This also reduces the memory requirements for the training set, as we can replace each high dimensional data point with its $k$ projection values (typically we use $10 \leq k \leq 20$).

We now define RP trees formally. For a cell containing points $S$, let $\Delta(S)$ be the diameter of $S$ (the distance between the two furthest points in the set), and $\Delta_A(S)$ the *average* diameter, that is, the

average distance between points of $S$:

$$\Delta_A^2(S) \;=\; \frac{1}{|S|^2} \sum_{x,y \in S} \|x - y\|^2 \;=\; \frac{2}{|S|} \sum_{x \in S} \|x - \text{mean}(S)\|^2.$$

We use two different types of splits: if $\Delta^2(S)$ is less than $c\Delta_A^2(S)$ (for some constant $c$) then we use the hyperplane split discussed above. Otherwise, we split $S$ into two groups based on distance from the mean.

**procedure** CHOOSERULE($S$)
  **comment:** RP tree version

**if** $\Delta^2(S) \le c \cdot \Delta_A^2(S)$
**then** $\begin{cases} \text{choose a random unit direction } v \\ \text{sort projection values: } a(x) = v \cdot x \;\; \forall x \in S, \text{ generating the list } a_1 \le a_2 \le \cdots \le a_n \\ \textbf{for } i = 1, \ldots, n-1 \text{ compute} \\ \quad \begin{cases} \mu_1 = \frac{1}{i}\sum_{j=1}^{i} a_j, \; \mu_2 = \frac{1}{n-i}\sum_{j=i+1}^{n} a_j \\ c_i = \sum_{j=1}^{i}(a_j - \mu_1)^2 + \sum_{j=i+1}^{n}(a_j - \mu_2)^2 \end{cases} \\ \text{find } i \text{ that minimizes } c_i \text{ and set } \theta = (a_i + a_{i+1})/2 \\ Rule(x) := v \cdot x \le \theta \end{cases}$
  **else** $\{Rule(x) := \|x - \text{mean}(S)\| \le \text{median}\{\|z - \text{mean}(S)\| : z \in S\}$
**return** ($Rule$)

In the first type of split, the data in a cell are projected onto a random direction and an appropriate split point is chosen. This point is not necessarily the median (as in $k$-d trees), but rather the position that maximally decreases average squared interpoint distance. In Figure 4.4, for instance, splitting the bottom cell at the median would lead to a messy partition, whereas the RP tree split produces two clean, connected clusters.

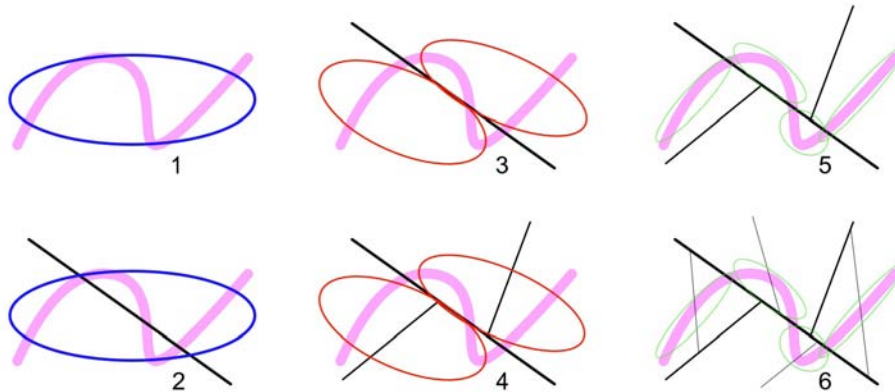

Figure 4: **An illustration of the RP-Tree algorithm.** 1: The full data set and the PCA ellipse that approximates it. 2: The first level split. 3: The two PCA ellipses corresponding to the two cells after the first split. 4: The two splits in the second level. 5: The four PCA ellipses for the cells at the third level. 6: The four splits at the third level. As the cells get smaller, their individual PCAs reveal 1D manifold structure. Note: the ellipses are for comparison only; the RP tree algorithm does not look at them.

The second type of split, based on distance from the mean of the cell, is needed to deal with cases in which the cell contains data at very different scales. In Figure 2, for instance, suppose that the vast majority of data is concentrated at the singleton "0-dimensional" point. If only splits by projection were allowed, then a large number of splits would be devoted to uselessly subdividing this point mass. The second type of split separates it from the rest of the data in one go. For a more concrete example, suppose that the data are image patches. A large fraction of them might be "empty" background patches, in which case they'd fall near the center of the cell in a very tight cluster. The

remaining image patches will be spread out over a much larger space. The effect of the split is then to separate out these two clusters.

## 2.3 Theoretical foundations

In analyzing RP trees, we consider a statistical notion of dimension: we say set $S$ has *local covariance dimension* $(d, \epsilon)$ if $(1 - \epsilon)$ fraction of the variance is concentrated in a $d$-dimensional subspace. To make this precise, start by letting $\sigma_1^2 \geq \sigma_2^2 \geq \cdots \geq \sigma_D^2$ denote the eigenvalues of the covariance matrix; these are the variances in each of the eigenvector directions.

**Definition 1** $S \subset \mathbb{R}^D$ *has local covariance dimension* $(d, \epsilon)$ *if the largest* $d$ *eigenvalues of its covariance matrix satisfy* $\sigma_1^2 + \cdots + \sigma_d^2 \geq (1 - \epsilon) \cdot (\sigma_1^2 + \cdots + \sigma_D^2)$. *(Note that* $\sigma_1^2 + \cdots + \sigma_D^2 = (1/2)\Delta_A^2(S)$.)

Now, suppose an RP tree is built from a data set $X \subset \mathbb{R}^D$, not necessarily finite. Recall that there are two different types of splits; let's call them splits *by distance* and splits *by projection*.

**Theorem 2** *There are constants* $0 < c_1, c_2, c_3 < 1$ *with the following property. Suppose an RP tree is built using data set* $X \subset \mathbb{R}^D$. *Consider any cell* $C$ *for which* $X \cap C$ *has local covariance dimension* $(d, \epsilon)$, *where* $\epsilon < c_1$. *Pick a point* $x \in S \cap C$ *at random, and let* $C'$ *be the cell that contains it at the next level down.*

- *If* $C$ *is split by distance then*

$$\mathbb{E}\left[\Delta(S \cap C')\right] \leq c_2 \Delta(S \cap C).$$

- *If* $C$ *is split by projection, then*

$$\mathbb{E}\left[\Delta_A^2(S \cap C')\right] \leq \left(1 - \frac{c_3}{d}\right) \Delta_A^2(S \cap C).$$

*In both cases, the expectation is over the randomization in splitting* $C$ *and the choice of* $x \in S \cap C$.

As a consequence, the expected average diameter of cells is halved every $O(d)$ levels. The proof of this theorem is in the supplementary material, along with even stronger results for different notions of dimension.

# 3 Experimental Results

## 3.1 A streaming version of the algorithm

The version of the RP algorithm we use in practice differs from the one above in three ways. First of all, both splits operate on the projected data; for the second type of split (split by distance), data that fall in an interval around the median are separated from data outside that interval. Second, the tree is built in a *streaming* manner: that is, the data arrive one at a time, and are processed (to update the tree) and immediately discarded. This is managed by maintaining simple statistics at each internal node of the tree and updating them appropriately as the data streams by (more details in the supplementary matter). The resulting efficiency is crucial to the large-scale applications we have in mind. Finally, instead of choosing a new random projection in each cell, a dictionary of a few random projections is chosen at the outset. In each cell, every one of these projections is tried out and the best one (that gives the largest decrease in $\Delta_A^2(S)$) is retained. This last step has the effect of boosting the probability of a good split.

## 3.2 Synthetic datasets

We start by considering two synthetic datasets that illustrate the shortcomings of $k$-d trees. We will see that RP trees adapt well to such cases. For the first dataset, points $x_1, \ldots, x_n \in \mathbb{R}^D$ are generated by the following process: for each point $x_i$,

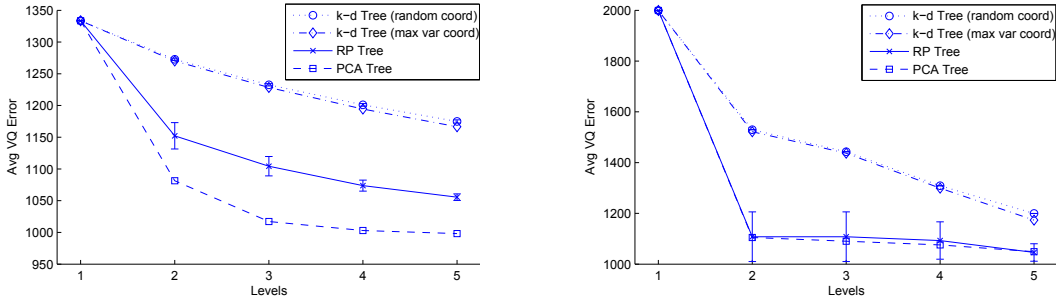

Figure 5: Performance of RP trees with $k$-d trees on first synthetic dataset (left) and the second synthetic dataset (right)

- choose $p_i$ uniformly at random from $[0, 1]$, and

- select each coordinate $x_{ij}$ independently from $N(p_i, 1)$.

For the second dataset, we choose $n$ points from two $D$-dimensional Gaussians (with equal probability) with means at $(-1, -1, \ldots, -1)$ and $(1, 1, \ldots, 1)$, and identity covariances.

We compare the performance of different trees according to the average VQ error they incur at various levels. We consider four types of trees: (1) $k$-d trees in which the coordinate for a split is chosen at random; (2) $k$-d trees in which at each split, the best coordinate is chosen (the one that most improves VQ error); (3) RP trees; and (4) for reference, PCA trees.

Figure 5 shows the results for the two datasets ($D = 1{,}000$ and $n = 10{,}000$) averaged over 15 runs. In both cases, RP trees outperform both $k$-d tree variants and are close to the performance of PCA trees without having to explicitly compute any principal components.

### 3.3 MNIST dataset

We next demonstrate RP trees on the all-familiar MNIST dataset of handwritten digits. This dataset consists of $28 \times 28$ grayscale images of the digits zero through nine, and is believed to have low intrinsic dimension (for instance, see [10]). We restrict our attention to digit 1 for this discussion.

Figure 6 (top) shows the first few levels of the RP tree for the images of digit 1. Each node is represented by the mean of the datapoints falling into that cell. Hence, the topmost node shows the mean of the entire dataset; its left and the right children show the means of the points belonging to their respective partitions, and so on. The bar underneath each node shows the fraction of points going to the left and to the right, to give a sense of how balanced each split is. Alongside each mean, we also show a histogram of the 20 largest eigenvalues of the covariance matrix, which reveal how closely the data in the cell is concentrated near a low-dimensional subspace. The last bar in the histogram is the variance unaccounted for.

Notice that most of the variance lies in a small number of directions, as might be expected. And this rapidly becomes more pronounced as we go further down in the tree. Hence, very quickly, the cell means become good representatives of the dataset: an experimental corroboration that RP trees adapt to the low intrinsic dimension of the data.

This is also brought out in Figure 6 (bottom), where the images are shown projected onto the plane defined by their top two principal components. (The outer ring of images correspond to the linear combinations of the two eigenvectors at those locations in the plane.) The left image shows how the data was split at the topmost level (dark versus light). Observe that this random cut is actually quite close to what the PCA split would have been, corroborating our earlier intuition (recall Figure 3). The right image shows the same thing, but for the first two levels of the tree: data is shown in four colors corresponding to the four different cells.

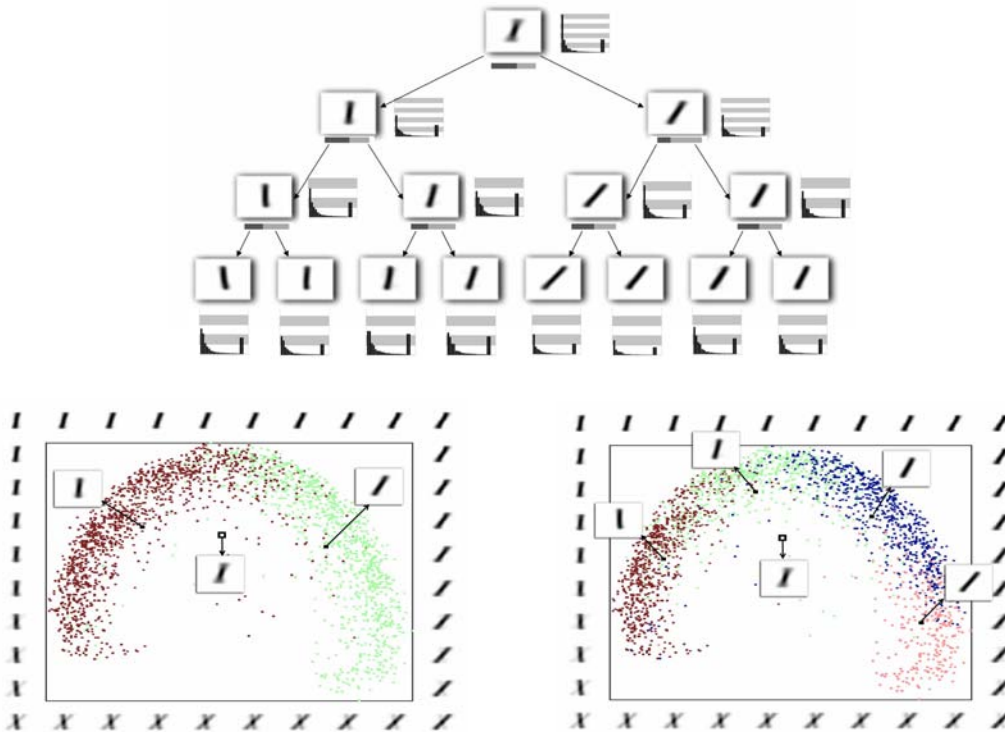

Figure 6: Top: Three levels of the RP tree for MNIST digit 1. Bottom: Images projected onto the first two principal components. Colors represent different cells in the RP tree, after just one split (left) or after two levels of the tree (right).

## Footnotes

* Corresponding author: `yfreund@cs.ucsd.edu`.

† Dasgupta and Verma acknowledge the support of NSF, under grants IIS-0347646 and IIS-0713540.

[1]This is in contrast to the *max diameter*, the maximum distance between two vectors in a cell.

[2]Here the probability is taken over the randomness in constructing the tree.

# References

[1] M. Belkin and P. Niyogi. Laplacian eigenmaps for dimensionality reduction and data representation. *Neural Computation*, 15(6):1373–1396, 2003.

[2] M. Belkin, P. Niyogi, and V. Sindhwani. On manifold regularization. *Conference on AI and Statistics*, 2005.

[3] J. Bentley. Multidimensional binary search trees used for associative searching. *Communications of the ACM*, 18(9):509–517, 1975.

[4] W. Boothby. *An Introduction to Differentiable Manifolds and Riemannian Geometry*. Academic Press, 2003.

[5] T. M. Cover and P. E. Hart. Nearest neighbor pattern classifications. *IEEE Transactions on Information Theory*, 13(1):21–27, 1967.

[6] M. Datar, N. Immorlica, P. Indyk, and V. Mirrokni. Locality sensitive hashing scheme based on p-stable distributions. *Symposium on Computational Geometry*, 2004.

[7] L. Devroye, L. Gyorfi, and G. Lugosi. *A Probabilistic Theory of Pattern Recognition*. Springer, 1996.

[8] T. Liu, A. Moore, A. Gray, and K. Yang. An investigation of practical approximate nearest neighbor algorithms. *Advances in Neural Information Processing Systems*, 2004.

[9] J. McNames. A fast nearest neighbor algorithm based on a principal axis search tree. *IEEE Transactions on Pattern Analysis and Machine Intelligence*, 23(9):964–976, 2001.

[10] M. Raginsky and S. Lazebnik. Estimation of intrinsic dimensionality using high-rate vector quantization. *Advances in Neural Information Processing Systems*, 18, 2006.

[11] S. Roweis and L. Saul. Nonlinear dimensionality reduction by locally linear embedding. *Science*, 290:2323–2326, 2000.

[12] J. Tenenbaum, V. de Silva, and J. Langford. A global geometric framework for nonlinear dimensionality reduction. *Science*, 290(5500):2319–2323, 2000.

